# Threshold Network Learning in the Presence of Equivalences

John Shawe-Taylor
Department of Computer Science
Royal Holloway and Bedford New College
University of London
Egham, Surrey TW20 0EX, UK

## Abstract

This paper applies the theory of Probably Approximately Correct (PAC) learning to multiple output feedforward threshold networks in which the weights conform to certain equivalences. It is shown that the sample size for reliable learning can be bounded above by a formula similar to that required for single output networks with no equivalences. The best previously obtained bounds are improved for all cases.

## 1  INTRODUCTION

This paper develops the results of Baum and Haussler [3] bounding the sample sizes required for reliable generalisation of a single output feedforward threshold network. They prove their result using the theory of Probably Approximately Correct (PAC) learning introduced by Valiant [11]. They show that for $0 < \epsilon \leq 1/2$, if a sample of size

$$m \geq m_0 = \frac{64W}{\epsilon} \log \frac{64N}{\epsilon}$$

is loaded into a feedforward network of linear threshold units with $N$ nodes and $W$ weights, so that a fraction $1 - \epsilon/2$ of the examples are correctly classified, then with confidence approaching certainty the network will correctly classify a fraction $1 - \epsilon$ of future examples drawn according to the same distribution. A similar bound was obtained for the case when the network correctly classified the whole sample. The results below will imply a significant improvement to both of these bounds.

In many cases training can be simplified if known properties of a problem can be incorporated into the structure of a network before training begins. One such technique is described by Shawe-Taylor [9], though many similar techniques have been applied as for example in TDNN's [6]. The effect of these restrictions is to constrain groups of weights to take the same value and learning algorithms are adapted to respect this constraint.

In this paper we consider the effect of this restriction on the generalisation performance of the networks and in particular the sample sizes required to obtain a given level of generalisation. This extends the work described above by Baum and Haussler [3] by improving their bounds and also improving the results of Shawe-Taylor and Anthony [10], who consider generalisation of multiple-output threshold networks. The remarkable fact is that in all cases the formula obtained is the same, where we now understand the number of weights $W$ to be the number of weight classes, but $N$ is still the number of computational nodes.

# 2    DEFINITIONS AND MAIN RESULTS

## 2.1    SYMMETRY AND EQUIVALENCE NETWORKS

We begin with a definition of threshold networks. To simplify the exposition it is convenient to incorporate the threshold value into the set of weights. This is done by creating a distinguished input that always has value 1 and is called the threshold input. The following is a formal notation for these systems.

A network $\mathcal{N} = (C, I, O, n_0, E)$ is specified by a set $C$ of computational nodes, a set $I$ of input nodes, a subset $O \subseteq C$ of output nodes and a node $n_0 \in I$, called the threshold node. The connectivity is given by a set $E \subseteq (C \cup I) \times C$ of connections, with $\{n_0\} \times C \subseteq E$.

With network $\mathcal{N}$ we associate a weight function $w$ from the set of connections to the real numbers. We say that the network $\mathcal{N}$ is in state $w$. For input vector $\mathbf{i}$ with values in some subset of the set $\mathcal{R}$ of real numbers, the network computes a function $F_{\mathcal{N}}(w, \mathbf{i})$.

An automorphism $\gamma$ of a network $\mathcal{N} = (C, I, O, n_0, E)$ is a bijection of the nodes of $\mathcal{N}$ which fixes $I$ setwise and $\{n_0\} \cup O$ pointwise, such that the induced action fixes $E$ setwise. We say that an automorphism $\gamma$ preserves the weight assignment $w$ if $w_{ji} = w_{(\gamma j)(\gamma i)}$ for all $i \in I \cup C$, $j \in C$. Let $\gamma$ be an automorphism of a network $\mathcal{N} = (C, I, O, n_0, E)$ and let $\mathbf{i}$ be an input to $\mathcal{N}$. We denote by $\mathbf{i}^\gamma$ the input whose value on input $k$ is that of $\mathbf{i}$ on input $\gamma^{-1} k$.

The following theorem is a natural generalisation of part of the Group Invariance Theorem of Minsky and Pappert [8] to multi-layer perceptrons.

**Theorem 2.1** *[9] Let $\gamma$ be a weight preserving automorphism of the network $\mathcal{N} = (C, I, O, n_0, E)$ in state $w$. Then for every input vector $\mathbf{i}$*

$$F_{\mathcal{N}}(w, \mathbf{i}) = F_{\mathcal{N}}(w, \mathbf{i}^\gamma).$$

Following this theorem it is natural to consider the concept of a *symmetry network* [9]. This is a pair $(\mathcal{N}, \Gamma)$, where $\mathcal{N}$ is a network and $\Gamma$ a group of weight

preserving automorphims of $\mathcal{N}$. We will also refer to the automorphisms as symmetries. For a symmetry network $(\mathcal{N}, \Gamma)$, we term the orbits of the connections $E$ under the action of $\Gamma$ the weight classes.

Finally we introduce the concept of an *equivalence network*. This definition abstracts from the symmetry networks precisely those properties we require to obtain our results. The class of equivalence networks is, however, far larger than that of symmetry networks and includes many classes of networks studied by other researchers [6, 7].

**Definition 2.2** *An equivalence network is a threshold network in which an equivalence relation is defined on both weights and nodes. The two relations are required to be compatible in that weights in the same class are connected to nodes in the same class, while nodes in the same class have the same set of input weight connection types. The weights in an equivalence class are at all times required to remain equal.*

Note that every threshold network can be viewed as an equivalence network by taking the trivial equivalence relations. We now show that symmetry networks are indeed equivalence networks with the same weight classes and give a further technical lemma. For both lemmas proofs are omitted.

**Lemma 2.3** *A symmetry network $(\mathcal{N}, \Gamma)$ is an equivalence network, where the equivalence classes are the orbits of connections and nodes respectively.*

**Lemma 2.4** *Let $\mathcal{N}$ be an equivalence network and $C$ be the set of classes of nodes. Then there is an indexing of the classes, $C_i$, $i = 1, \ldots, n$, such that nodes in $C_i$ do not have connections from nodes in $C_j$ for $j \geq i$.*

## 2.2  MAIN RESULTS

We are now in a position to state our main results. Note that throughout this paper log means natural logarithm, while an explicit subscript is used for other bases.

**Theorem 2.5** *Let $\mathcal{N}$ be an equivalence network with $W$ weight classes and $N$ computational nodes. If the network correctly computes a function on a set of $m$ inputs drawn independently according to a fixed probability distribution, where*

$$m \geq m_0(\epsilon, \delta) = \frac{1}{\epsilon(1 - \sqrt{\epsilon})} \left[ \log\left(\frac{1.3}{\delta}\right) + 2W \log\left(\frac{6\sqrt{N}}{\epsilon}\right) \right]$$

*then with probability at least $1 - \delta$ the error rate of the network will be less than $\epsilon$ on inputs drawn according to the same distribution.*

**Theorem 2.6** *Let $\mathcal{N}$ be an equivalence network with $W$ weight classes and $N$ computational nodes. If the network correctly computes a function on a fraction $1 - (1 - \gamma)\epsilon$ of $m$ inputs drawn independently according to a fixed probability distribution, where*

$$m \geq m_0(\epsilon, \delta, \gamma) = \frac{1}{\gamma^2 \epsilon (1 - \sqrt{\epsilon/N})} \left[ 4 \log\left(\frac{4}{\delta}\right) + 6W \log\left(\frac{4N}{\gamma^{2/3}\epsilon}\right) \right]$$

*then with probability at least $1 - \delta$ the error rate of the network will be less than $\epsilon$ on inputs drawn according to the same distribution.*

# 3   THEORETICAL BACKGROUND

## 3.1   DEFINITIONS AND PREVIOUS RESULTS

In order to present results for binary outputs ($\{0, 1\}$ functions) and larger ranges in a unified way we will consider throughout the task of learning the graph of a function. All the definitions reduce to the standard ones when the outputs are binary.

We consider learning from examples as selecting a suitable function from a set $H$ of hypotheses, being functions from a space $X$ to set $Y$, which has at most countable size. At all times we consider an (unknown) target function

$$c : X \longrightarrow Y$$

which we are attempting to learn. To this end the space $X$ is required to be a probability space $(X, \Sigma, \mu)$, with appropriate regularity conditions so that the sets considered are measurable [4]. In particular the hypotheses should be measurable when $Y$ is given the discrete topology as should the error sets defined below. The space $S = X \times Y$ is equipped with a $\sigma$-algebra $\Sigma \times 2^Y$ and measure $\nu = \nu(\mu, c)$, defined by its value on sets of the form $U \times \{y\}$:

$$\nu(U \times \{y\}) = \mu\left(U \cap c^{-1}(y)\right).$$

Using this measure the error of a hypothesis is defined to be

$$\mathrm{er}_\nu(h) = \nu\{(x, y) \in S | h(x) \neq y\}.$$

The introduction of $\nu$ allows us to consider samples being drawn from $S$, as they will automatically reflect the output value of the target. This approach freely generalises to stochastic concepts though we will restrict ourselves to target functions for the purposes of this paper. The error of a hypothesis $h$ on a sample $\mathbf{x} = ((x_1, y_1), \ldots, (x_m, y_m)) \in S^m$ is defined to be

$$\mathrm{er}_\mathbf{x}(h) = \frac{1}{m} |\{i | h(x_i) \neq y_i\}|.$$

We also define the VC dimension of a set of hypotheses by reference to the product space $S$. Consider a sample $\mathbf{x} = ((x_1, y_1), \ldots, (x_m, y_m)) \in S^m$ and the function

$$\mathbf{x}^\star : H \longrightarrow \{0, 1\}^m,$$

given by $\mathbf{x}^\star(h)_i = 1$ if and only if $h(x_i) = y_i$, for $i = 1, \ldots, m$. We can now define the growth function $B_H(m)$ as

$$B_H(m) = \max_{\mathbf{x} \in S^m} |\{\mathbf{x}^\star(h) | h \in H\}| \leq 2^m.$$

The Vapnik-Chervonenkis dimension of a hypothesis space $H$ is defined as

$$\mathrm{VCdim}(H) = \begin{cases} \infty; & \text{if } B_H(m) = 2^m, \text{ for all } m; \\ \max\{m | B_H(m) = 2^m\}; & \text{otherwise.} \end{cases}$$

In the case of a threshold network $\mathcal{N}$, the set of functions obtainable using all possible weight assignments is termed the hypothesis space of $\mathcal{N}$ and we will refer

to it as $\mathcal{N}$. For a threshold network $\mathcal{N}$, we also introduce the state growth function $S_{\mathcal{N}}(m)$. This is defined by first considering all computational nodes to be output nodes, and then counting different output sequences.

$$S_{\mathcal{N}}(m) = \max_{\mathbf{x}=(\mathbf{i}_1,\ldots,\mathbf{i}_m)\in X^m} |\{(F_{\mathcal{N}'}(w,\mathbf{i}_1), F_{\mathcal{N}'}(w,\mathbf{i}_2),\ldots, F_{\mathcal{N}'}(w,\mathbf{i}_m))|w : E \to \mathcal{R}\}|$$

where $X = [0,1]^{|I|}$ and $\mathcal{N}'$ is obtained from $\mathcal{N}$ by setting $O = C$. We clearly have that for all $\mathcal{N}$ and $m$, $B_{\mathcal{N}}(m) \le S_{\mathcal{N}}(m)$.

**Theorem 3.1** *[2] If a hypothesis space $H$ has growth function $B_H(m)$ then for any $\epsilon > 0$ and $k > m$ and*

$$0 < r < 1 - \frac{1}{\sqrt{\epsilon k}}$$

*the probability that there is a function in $H$ which agrees with a randomly chosen $m$ sample and has error greater than $\epsilon$ is less than*

$$\frac{\epsilon k(1-r)^2}{\epsilon k(1-r)^2 - 1} B_H(m+k) \exp\left\{-r\epsilon \frac{km}{m+k}\right\}.$$

This result can be used to obtain the following bound on sample size required for PAC learnability of a hypothesis space with VC dimension $d$. The theorem improves the bounds reported by Blumer et al. [4].

**Theorem 3.2** *[2] If a hypothesis space $H$ has finite VC dimension $d > 1$, then there is $m_0 = m_0(\epsilon, \delta)$ such that if $m > m_0$ then the probability that a hypothesis consistent with a randomly chosen sample of size $m$ has error greater than $\epsilon$ is less than $\delta$. A suitable value of $m_0$ is*

$$m_0 = \frac{1}{\epsilon(1-\sqrt{\epsilon})}\left[\log\left(\frac{d/(d-1)}{\delta}\right) + 2d\log\left(\frac{6}{\epsilon}\right)\right].$$

$\square$

For the case when we allow our hypothesis to incorrectly compute the function on a small fraction of the training sample, we have the following result. Note that we are still considering the discrete metric and so in the case where we are considering multiple output feedforward networks a single output in error would count as an overall error.

**Theorem 3.3** *[10] Let $0 < \epsilon < 1$ and $0 < \gamma \le 1$. Suppose $H$ is a hypothesis space of functions from an input space $X$ to a possibly countable set $Y$, and let $\nu$ be any probability measure on $S = X \times Y$. Then the probability (with respect to $\nu^m$) that, for $\mathbf{x} \in S^m$, there is some $h \in H$ such that*

$$\mathrm{er}_{\nu}(h) > \epsilon \quad \text{and} \quad \mathrm{er}_{\mathbf{x}}(h) \le (1-\gamma)\mathrm{er}_{\nu}(h)$$

*is at most*

$$4\,B_H(2m) \exp\left(-\frac{\gamma^2 \epsilon m}{4}\right).$$

*Furthermore, if $H$ has finite VC dimension $d$, this quantity is less than $\delta$ for*

$$m > m_0(\epsilon, \delta, \gamma) = \frac{1}{\gamma^2\epsilon(1-\sqrt{\epsilon})}\left[4\log\left(\frac{4}{\delta}\right) + 6d\log\left(\frac{4}{\gamma^{2/3}\epsilon}\right)\right].$$

$\square$

## 4    THE GROWTH FUNCTION FOR EQUIVALENCE NETWORKS

We will bound the number of output sequences $B_{\mathcal{N}}(m)$ for a number $m$ of inputs by the number of distinct state sequences $S_{\mathcal{N}}(m)$ that can be generated from the $m$ inputs by different weight assignments. This follows the approach taken in [10].

**Theorem 4.1** *Let $\mathcal{N}$ be an equivalence network with $W$ weight equivalence classes and a total of $N$ computational nodes. Then we can bound $S_{\mathcal{N}}(m)$ by*

$$S_{\mathcal{N}}(m) \leq \left(\frac{Nem}{W}\right)^W.$$

**Idea of Proof:** Let $C_i$, $i = 1, \ldots, n$, be the equivalence classes of nodes indexed as guaranteed by Lemma 2.4 with $|C_i| = c_i$ and the number of inputs for nodes in $C_i$ being $n_i$ (including the threshold input). Denote by $\mathcal{N}_j$ the network obtained by taking only the first $j$ node equivalence classes. We omit a proof by induction that

$$S_{\mathcal{N}_j}(m) \leq \prod_{i=1}^{j} B_i(mc_i),$$

where $B_i$ is the growth function for nodes in the class $C_i$.

Using the well known bound on the growth function of a threshold node with $n_i$ inputs we obtain

$$S_{\mathcal{N}}(m) \leq \prod_{i=1}^{n} \left(\frac{emc_i}{n_i}\right)^{n_i}.$$

Consider the function $f(x) = x \log x$. This is a convex function and so for a set of values $x_1, \ldots, x_M$, we have that the average of $f(x_i)$ is greater than or equal to $f$ applied to the average of $x_i$. Consider taking the $x$'s to be $c_i$ copies of $n_i/c_i$ for each $i = 1, \ldots n$. We obtain

$$\frac{1}{N}\sum_{i=1}^{n} n_i \log \frac{n_i}{c_i} \geq \frac{W}{N}\log\frac{W}{N} \quad \text{or} \quad \prod_{i=1}^{n}\left(\frac{c_i}{n_i}\right)^{n_i} \leq \left(\frac{N}{W}\right)^W,$$

and so

$$S_{\mathcal{N}}(m) \leq \left(\frac{emN}{W}\right)^W,$$

as required. ∎

The bounds we have obtained make it possible to bound the Vapnik-Chervonenkis dimension of equivalence networks. Though we we will not need these results, we give them here for completeness.

**Proposition 4.2** *The Vapnik-Chervonenkis dimension of an equivalence network with $W$ weight classes and $N$ computational nodes is bounded by*

$$2W \log_2 eN.$$

# 5   PROOF OF MAIN RESULTS

Using the results of the last section we are now in a position to prove Theorems 2.5 and 2.6.

**Proof of Theorem 2.5:** (Outline) We use Theorem 3.1 which bounds the probability that a hypothesis with error greater than $\epsilon$ can match an $m$-sample. Substituting our bound on the growth function of an equivalence network and choosing

$$k = \left\lceil m \left( \frac{r\epsilon m}{W} - 1 \right) \right\rceil,$$

and $r$ as in [1], we obtain the following bound on the probability

$$\left( \frac{d}{d-1} \right) \left( \frac{e^4 \epsilon m^2}{W^2} \right)^W N^W \exp(-\epsilon m).$$

By choosing $m > m_0$ where $m_0$ is given by

$$m_0 = m_0(\epsilon, \delta) = \frac{1}{\epsilon(1 - \sqrt{\epsilon})} \left[ \log \left( \frac{1.3}{\delta} \right) + 2W \log \left( \frac{6\sqrt{N}}{\epsilon} \right) \right]$$

we guarantee that the above probability is less than $\delta$ as required. ∎

Our second main result can be obtained more directly.

**Proof of Theorem 2.6:** (Outline) We use Theorem 3.3 which bounds the probability that a hypothesis with error greater than $\epsilon$ can match all but a fraction $(1 - \gamma)$ of an $m$-sample. The bound on the sample size is obtained from the probability bound by using the inequality for $B_H(2m)$. By adjusting the parameters we will convert the probability expression to that obtained by substituting our growth function. We can then read off a sample size by the corresponding substitution in the sample size formula. Consider setting $d = W$, $\epsilon = \epsilon'/N$ and $m = Nm'$. With these substitutions the sample size formula is

$$m' = \frac{1}{\gamma^2 \epsilon' (1 - \sqrt{\epsilon'/N})} \left[ 4 \log \left( \frac{4}{\delta} \right) + 6W \log \left( \frac{4N}{\gamma^{2/3} \epsilon'} \right) \right]$$

as required. ∎

# 6   CONCLUSION

The problem of training feedforward neural networks remains a major hurdle to the application of this approach to large scale systems. A very promising technique for simplifying the training problem is to include equivalences in the network structure which can be justified by a priori knowledge of the application domain. This paper has extended previous results concerning sample sizes for feedforward networks to cover so called equivalence networks in which weights are constrained in this way. At the same time we have improved the sample size bounds previously obtained for standard threshold networks [3] and multiple output networks [10].

The results are of the same order as previous results and imply similar bounds on the Vapnik-Chervonenkis namely $2W \log_2 eN$. They perhaps give circumstancial evidence for the conjecture that the $\log_2 eN$ factor in this expression is real, in that the same expression obtains even if the number of computational nodes is increased by expanding the equivalence classes of weights. Equivalence networks may be a useful area to search for high growth functions and perhaps show that for certain classes the VC dimension is $\Omega(W \log N)$.

# References

[1] Martin Anthony, Norman Biggs and John Shawe-Taylor, Learnability and Formal Concept Analysis, RHBNC Department of Computer Science, Technical Report, CSD-TR-624, 1990.

[2] Martin Anthony, Norman Biggs and John Shawe-Taylor, The learnability of formal concepts, Proc. COLT '90, Rochester, NY. (eds Mark Fulk and John Case) (1990) 246-257.

[3] Eric Baum and David Haussler, What size net gives valid generalization, Neural Computation, 1 (1) (1989) 151-160.

[4] Anselm Blumer, Andrzej Ehrenfeucht, David Haussler and Manfred K. Warmuth, Learnability and the Vapnik-Chervonenkis dimension, JACM, 36 (4) (1989) 929-965.

[5] David Haussler, preliminary extended abstract, COLT '89.

[6] K. Lang and G.E. Hinton, The development of TDNN architecture for speech recognition, Technical Report CMU-CS-88-152, Carnegie-Mellon University, 1988.

[7] Y. le Cun, A theoretical framework for back propagation, in D. Touretzsky, editor, *Connectionist Models: A Summer School*, Morgan-Kaufmann, 1988.

[8] M. Minsky and S. Papert, Perceptrons, expanded edition, MIT Press, Cambridge, USA, 1988.

[9] John Shawe-Taylor, Building Symmetries into Feedforward Network Architectures, Proceedings of First IEE Conference on Artificial Neural Networks, London, 1989, 158-162.

[10] John Shawe-Taylor and Martin Anthony, Sample Sizes for Multiple Output Feedforward Networks, Network, 2 (1991) 107-117.

[11] Leslie G. Valiant, A theory of the learnable, Communications of the ACM, 27 (1984) 1134-1142.
